# Computing with infinite networks

**Christopher K. I. Williams**
Neural Computing Research Group
Department of Computer Science and Applied Mathematics
Aston University, Birmingham B4 7ET, UK
`c.k.i.williams@aston.ac.uk`

## Abstract

For neural networks with a wide class of weight-priors, it can be shown that in the limit of an infinite number of hidden units the prior over functions tends to a Gaussian process. In this paper analytic forms are derived for the covariance function of the Gaussian processes corresponding to networks with sigmoidal and Gaussian hidden units. This allows predictions to be made efficiently using networks with an infinite number of hidden units, and shows that, somewhat paradoxically, it may be easier to compute with infinite networks than finite ones.

## 1 Introduction

To someone training a neural network by maximizing the likelihood of a finite amount of data it makes no sense to use a network with an infinite number of hidden units; the network will "overfit" the data and so will be expected to generalize poorly. However, the idea of selecting the network size depending on the amount of training data makes little sense to a Bayesian; a model should be chosen that reflects the understanding of the problem, and then application of Bayes' theorem allows inference to be carried out (at least in theory) after the data is observed.

In the Bayesian treatment of neural networks, a question immediately arises as to how many hidden units are believed to be appropriate for a task. Neal (1996) has argued compellingly that for real-world problems, there is no reason to believe that neural network models should be limited to nets containing only a "small" number of hidden units. He has shown that it is sensible to consider a limit where the number of hidden units in a net tends to infinity, and that good predictions can be obtained from such models using the Bayesian machinery. He has also shown that for fixed hyperparameters, a large class of neural network models will converge to a Gaussian process prior over functions in the limit of an infinite number of hidden units.

Neal's argument is an existence proof—it states that an infinite neural net will converge to a Gaussian process, but does not give the covariance function needed to actually specify the particular Gaussian process. In this paper I show that for certain weight priors and transfer functions in the neural network model, the covariance function which describes the behaviour of the corresponding Gaussian process can be calculated analytically. This allows predictions to be made using neural networks with an infinite number of hidden units in time $O(n^3)$, where $n$ is the number of training examples[1]. The only alternative currently available is to use Markov Chain Monte Carlo (MCMC) methods (e.g. Neal, 1996) for networks with a large (but finite) number of hidden units. However, this is likely to be computationally expensive, and we note possible concerns over the time needed for the Markov chain to reach equilibrium. The availability of an analytic form for the covariance function also facilitates the comparison of the properties of neural networks with an infinite number of hidden units as compared to other Gaussian process priors that may be considered.

The Gaussian process analysis applies for fixed hyperparameters $\theta$. If it were desired to make predictions based on a hyperprior $P(\theta)$ then the necessary $\theta$-space integration could be achieved by MCMC methods. The great advantage of integrating out the weights analytically is that it dramatically reduces the dimensionality of the MCMC integrals, and thus improves their speed of convergence.

## 1.1   From priors on weights to priors on functions

Bayesian neural networks are usually specified in a hierarchical manner, so that the weights $w$ are regarded as being drawn from a distribution $P(w|\theta)$. For example, the weights might be drawn from a zero-mean Gaussian distribution, where $\theta$ specifies the variance of groups of weights. A full description of the prior is given by specifying $P(\theta)$ as well as $P(w|\theta)$. The hyperprior can be integrated out to give $P(w) = \int P(w|\theta)P(\theta)\,d\theta$, but in our case it will be advantageous not to do this as it introduces weight correlations which prevent convergence to a Gaussian process.

In the Bayesian view of neural networks, predictions for the output value $y_*$ corresponding to a new input value $x_*$ are made by integrating over the posterior in weight space. Let $\mathcal{D} = ((x_1,t_1),(x_2,t_2),\ldots,(x_n,t_n))$ denote the $n$ training data pairs, $t = (t_1,\ldots,t_n)^T$ and $f_*(w)$ denote the mapping carried out by the network on input $x_*$ given weights $w$. $P(w|t,\theta)$ is the weight posterior given the training data[2]. Then the predictive distribution for $y_*$ given the training data and hyperparameters $\theta$ is

$$P(y_*|t,\theta) = \int \delta(y_* - f_*(w))P(w|t,\theta)\,dw \qquad (1)$$

We will now show how this can also be viewed as making the prediction using priors over functions rather than weights. Let $f(w)$ denote the vector of outputs corresponding to inputs $(x_1,\ldots,x_n)$ given weights $w$. Then, using Bayes' theorem we have $P(w|t,\theta) = P(t|w)P(w|\theta)/P(t|\theta)$, and $P(t|w) = \int P(t|y)\,\delta(y - f(w))\,dy$. Hence equation 1 can be rewritten as

$$P(y_*|t,\theta) = \frac{1}{P(t|\theta)}\int\int P(t|y)\,\delta(y_* - f_*(w))\delta(y - f(w))\,P(w|\theta)\,dw\,dy \quad (2)$$

However, the prior over $(y_*,y_1,\ldots,y_n)$ is given by $P(y_*,y|\theta) = P(y_*|y,\theta)P(y|\theta) = \int \delta(y_* - f_*(w))\,\delta(y - f(w))P(w|\theta)\,dw$ and thus the predictive distribution can be

written as

$$P(y_*|t, \boldsymbol{\theta}) = \frac{1}{P(t|\boldsymbol{\theta})} \int P(t|\boldsymbol{y})P(y_*|\boldsymbol{y}, \boldsymbol{\theta})P(\boldsymbol{y}|\boldsymbol{\theta}) \, d\boldsymbol{y} = \int P(y_*|\boldsymbol{y}, \boldsymbol{\theta})P(\boldsymbol{y}|t, \boldsymbol{\theta}) \, d\boldsymbol{y}$$

$$(3)$$

Hence in a Bayesian view it is the prior over function values $P(y_*, \boldsymbol{y}|\boldsymbol{\theta})$ which is important; specifying this prior by using weight distributions is one valid way to achieve this goal. In general we can use the weight space or function space view, which ever is more convenient, and for infinite neural networks the function space view is more useful.

## 2   Gaussian processes

A stochastic process is a collection of random variables $\{Y(\boldsymbol{x})|\boldsymbol{x} \in X\}$ indexed by a set $X$. In our case $X$ will be $\mathcal{R}^d$, where $d$ is the number of inputs. The stochastic process is specified by giving the probability distribution for every finite subset of variables $Y(\boldsymbol{x}_1), \ldots, Y(\boldsymbol{x}_k)$ in a consistent manner. A Gaussian process (GP) is a stochastic process which can be fully specified by its mean function $\mu(\boldsymbol{x}) = E[Y(\boldsymbol{x})]$ and its covariance function $C(\boldsymbol{x}, \boldsymbol{x}') = E[(Y(\boldsymbol{x}) - \mu(\boldsymbol{x}))(Y(\boldsymbol{x}') - \mu(\boldsymbol{x}'))]$; any finite set of $Y$-variables will have a joint multivariate Gaussian distribution. For a multidimensional input space a Gaussian process may also be called a Gaussian random field.

Below we consider Gaussian processes which have $\mu(\boldsymbol{x}) \equiv 0$, as is the case for the neural network priors discussed in section 3. A non-zero $\mu(\boldsymbol{x})$ can be incorporated into the framework at the expense of a little extra complexity.

A widely used class of covariance functions is the stationary covariance functions, whereby $C(\boldsymbol{x}, \boldsymbol{x}') = C(\boldsymbol{x} - \boldsymbol{x}')$. These are related to the spectral density (or power spectrum) of the process by the Wiener-Khinchine theorem, and are particularly amenable to Fourier analysis as the eigenfunctions of a stationary covariance kernel are $\exp i\boldsymbol{k}.\boldsymbol{x}$. Many commonly used covariance functions are also isotropic, so that $C(\boldsymbol{h}) = C(h)$ where $\boldsymbol{h} = \boldsymbol{x} - \boldsymbol{x}'$ and $h = |\boldsymbol{h}|$. For example $C(h) = \exp(-(h/\sigma)^\nu)$ is a valid covariance function for all $d$ and for $0 < \nu \leq 2$. Note that in this case $\sigma$ sets the correlation length-scale of the random field, although other covariance functions (e.g. those corresponding to power-law spectral densities) may have no preferred length scale.

### 2.1   Prediction with Gaussian processes

The model for the observed data is that it was generated from the *prior* stochastic process, and that independent Gaussian noise (of variance $\sigma_\nu^2$) was then added. Given a prior covariance function $C_P(\boldsymbol{x}_i, \boldsymbol{x}_j)$, a noise process $C_N(\boldsymbol{x}_i, \boldsymbol{x}_j) = \sigma_\nu^2 \delta_{ij}$ (i.e. independent noise of variance $\sigma_\nu^2$ at each data point) and the training data, the prediction for the distribution of $y_*$ corresponding to a test point $\boldsymbol{x}_*$ is obtained simply by applying equation 3. As the prior and noise model are both Gaussian the integral can be done analytically and $P(y_*|t, \boldsymbol{\theta})$ is Gaussian with mean and variance

$$\hat{y}(\boldsymbol{x}_*) = \boldsymbol{k}_P^T(\boldsymbol{x}_*)(K_P + K_N)^{-1}\boldsymbol{t} \tag{4}$$

$$\sigma_{\hat{y}}^2(\boldsymbol{x}_*) = C_P(\boldsymbol{x}_*, \boldsymbol{x}_*) - \boldsymbol{k}_P^T(\boldsymbol{x}_*)(K_P + K_N)^{-1}\boldsymbol{k}_P(\boldsymbol{x}_*) \tag{5}$$

where $[K_\alpha]_{ij} = C_\alpha(\boldsymbol{x}_i, \boldsymbol{x}_j)$ for $\alpha = P, N$ and $\boldsymbol{k}_P(\boldsymbol{x}_*) = (C_P(\boldsymbol{x}_*, \boldsymbol{x}_1), \ldots, C_P(\boldsymbol{x}_*, \boldsymbol{x}_n))^T$. $\sigma_{\hat{y}}^2(\boldsymbol{x}_*)$ gives the "error bars" of the prediction.

Equations 4 and 5 are the analogue for spatial processes of Wiener-Kolmogorov prediction theory. They have appeared in a wide variety of contexts including

geostatistics where the method is known as "kriging" (Journel and Huijbregts, 1978; Cressie 1993), multidimensional spline smoothing (Wahba, 1990), in the derivation of radial basis function neural networks (Poggio and Girosi, 1990) and in the work of Whittle (1963).

## 3  Covariance functions for Neural Networks

Consider a network which takes an input $x$, has one hidden layer with $H$ units and then linearly combines the outputs of the hidden units with a bias to obtain $f(x)$. The mapping can be written

$$f(x) = b + \sum_{j=1}^{H} v_j h(x; u_j) \tag{6}$$

where $h(x; u)$ is the hidden unit transfer function (which we shall assume is bounded) which depends on the input-to-hidden weights $u$. This architecture is important because it has been shown by Hornik (1993) that networks with one hidden layer are universal approximators as the number of hidden units tends to infinity, for a wide class of transfer functions (but excluding polynomials). Let $b$ and the $v$'s have independent zero-mean distributions of variance $\sigma_b^2$ and $\sigma_v^2$ respectively, and let the weights $u_j$ for each hidden unit be independently and identically distributed. Denoting all weights by $w$, we obtain (following Neal, 1996)

$$E_w[f(x)] = 0 \tag{7}$$

$$E_w[f(x)f(x')] = \sigma_b^2 + \sum_j \sigma_v^2 E_u[h_j(x; u)h_j(x'; u)] \tag{8}$$

$$= \sigma_b^2 + H\sigma_v^2 E_u[h(x; u)h(x'; u)] \tag{9}$$

where equation 9 follows because all of the hidden units are identically distributed. The final term in equation 9 becomes $\omega^2 E_u[h(x; u)h(x'; u)]$ by letting $\sigma_v^2$ scale as $\omega^2/H$.

As the transfer function is bounded, all moments of the distribution will be bounded and hence the Central Limit Theorem can be applied, showing that the stochastic process will become a Gaussian process in the limit as $H \to \infty$.

By evaluating $E_u[h(x)h(x')]$ for all $x$ and $x'$ in the training and testing sets we can obtain the covariance function needed to describe the neural network as a Gaussian process. These expectations are, of course, integrals over the relevant probability distributions of the biases and input weights. In the following sections two specific choices for the transfer functions are considered, (1) a sigmoidal function and (2) a Gaussian. Gaussian weight priors are used in both cases.

It is interesting to note why this analysis cannot be taken a stage further to integrate out any hyperparameters as well. For example, the variance $\sigma_v^2$ of the $v$ weights might be drawn from an inverse Gamma distribution. In this case the distribution $P(v) = \int P(v|\sigma_v^2)P(\sigma_v^2)d\sigma_v^2$ is no longer the product of the marginal distributions for each $v$ weight (in fact it will be a multivariate $t$-distribution). A similar analysis can be applied to the $u$ weights with a hyperprior. The effect is to make the hidden units non-independent, so that the Central Limit Theorem can no longer be applied.

### 3.1  Sigmoidal transfer function

A sigmoidal transfer function is a very common choice in neural networks research; nets with this architecture are usually called multi-layer perceptrons.

Below we consider the transfer function $h(x; u) = \Phi(u_0 + \sum_{i=1}^{d} u_j x_i)$, where $\Phi(z) = 2/\sqrt{\pi} \int_0^z e^{-t^2} dt$ is the error function, closely related to the cumulative distribution function for the Gaussian distribution. Appropriately scaled, the graph of this function is very similar to the tanh function which is more commonly used in the neural networks literature.

In calculating $V(x, x') \stackrel{def}{=} E_u[h(x; u)h(x'; u)]$ we make the usual assumptions (e.g. MacKay, 1992) that $u$ is drawn from a zero-mean Gaussian distribution with covariance matrix $\Sigma$, i.e. $u \sim N(0, \Sigma)$. Let $\tilde{x} = (1, x_1, \ldots, x_d)$ be an augmented input vector whose first entry corresponds to the bias. Then $V_{\text{erf}}(x, x')$ can be written as

$$V_{\text{erf}}(x, x') = \frac{1}{(2\pi)^{\frac{d+1}{2}} |\Sigma|^{1/2}} \int \Phi(u^T \tilde{x}) \Phi(u^T \tilde{x}') \exp(-\frac{1}{2} u^T \Sigma^{-1} u) \, du \qquad (10)$$

This integral can be evaluated analytically[3] to give

$$V_{\text{erf}}(x, x') = \frac{2}{\pi} \sin^{-1} \frac{2 \tilde{x}^T \Sigma \tilde{x}'}{\sqrt{(1 + 2\tilde{x}^T \Sigma \tilde{x})(1 + 2\tilde{x}'^T \Sigma \tilde{x}')}} \qquad (11)$$

We observe that this covariance function is not stationary, which makes sense as the distributions for the weights are centered about zero, and hence translational symmetry is not present.

Consider a diagonal weight prior so that $\Sigma = \text{diag}(\sigma_0^2, \sigma_f^2, \ldots, \sigma_f^2)$, so that the inputs $i = 1, \ldots, d$ have a different weight variance to the bias $\sigma_0^2$. Then for $|x|^2$, $|x'|^2 \gg (1 + 2\sigma_0^2)/2\sigma_f^2$, we find that $V_{\text{erf}}(x, x') \simeq 1 - 2\theta/\pi$, where $\theta$ is the angle between $x$ and $x'$. Again this makes sense intuitively; if the model is made up of a large number of sigmoidal functions in random directions (in $x$ space), then we would expect points that lie diametrically opposite (i.e. at $x$ and $-x$) to be anti-correlated, because they will lie in the $+1$ and $-1$ regions of the sigmoid function for most directions.

### 3.2 Gaussian transfer function

One other very common transfer function used in neural networks research is the Gaussian, so that $h(x; u) = \exp[-(x - u)^T(x - u)/2\sigma_g^2]$, where $\sigma_g^2$ is the width parameter of the Gaussian. Gaussian basis functions are often used in Radial Basis Function (RBF) networks (e.g. Poggio and Girosi, 1990).

For a Gaussian prior over the distribution of $u$ so that $u \sim N(0, \sigma_u^2 I)$,

$$V_G(x, x') = \frac{1}{(2\pi\sigma_u^2)^{d/2}} \int \exp -\frac{(x - u)^T(x - u)}{2\sigma_g^2} \exp -\frac{(x' - u)^T(x' - u)}{2\sigma_g^2} \exp -\frac{u^T u}{2\sigma_u^2} c \qquad (12)$$

By completing the square and integrating out $u$ we obtain

$$V_G(x, x') = \left(\frac{\sigma_e}{\sigma_u}\right)^d \exp\{-\frac{x^T x}{2\sigma_m^2}\} \exp\{-\frac{(x - x')^T(x - x')}{2\sigma_s^2}\} \exp\{-\frac{x'^T x'}{2\sigma_m^2}\} \qquad (13)$$

where $1/\sigma_e^2 = 2/\sigma_g^2 + 1/\sigma_u^2$, $\sigma_s^2 = 2\sigma_g^2 + \sigma_g^4/\sigma_u^2$ and $\sigma_m^2 = 2\sigma_u^2 + \sigma_g^2$. This formula can be generalized by allowing covariance matrices $\Sigma_b$ and $\Sigma_u$ in place of $\sigma_g^2 I$ and $\sigma_u^2 I$; rescaling each input variable $x_i$ independently is a simple example.

Again this is a non-stationary covariance function, although it is interesting to note that if $\sigma_u^2 \to \infty$ (while scaling $\omega^2$ appropriately) we find that $V_G(\boldsymbol{x}, \boldsymbol{x}') \propto \exp\{-(\boldsymbol{x} - \boldsymbol{x}')^T(\boldsymbol{x} - \boldsymbol{x}')/4\sigma_g^2\}$ [4]. For a finite value of $\sigma_u^2$, $V_G(\boldsymbol{x}, \boldsymbol{x}')$ is a stationary covariance function "modulated" by the Gaussian decay function $\exp(-\boldsymbol{x}^T\boldsymbol{x}/2\sigma_m^2)\exp(-\boldsymbol{x}'^T\boldsymbol{x}'/2\sigma_m^2)$. Clearly if $\sigma_m^2$ is much larger than the largest distance in $\boldsymbol{x}$-space then the predictions made with $V_G$ and a Gaussian process with only the stationary part of $V_G$ will be very similar.

It is also possible to view the infinite network with Gaussian transfer functions as an example of a shot-noise process based on an inhomogeneous Poisson process (see Parzen (1962) §4.5 for details). Points are generated from an inhomogeneous Poisson process with the rate function $\propto \exp(-\boldsymbol{x}^T\boldsymbol{x}/2\sigma_u^2)$, and Gaussian kernels of height $v$ are centered on each of the points, where $v$ is chosen iid from a distribution with mean zero and variance $\sigma_v^2$.

### 3.3  Comparing covariance functions

The priors over functions specified by sigmoidal and Gaussian neural networks differ from covariance functions that are usually employed in the literature, e.g. splines (Wahba, 1990). How might we characterize the different covariance functions and compare the kinds of priors that they imply ?

The complex exponential $\exp i\boldsymbol{k}.\boldsymbol{x}$ is an eigenfunction of a stationary and isotropic covariance function, and hence the spectral density (or power spectrum) $S(k)$ ($k = |\boldsymbol{k}|$) nicely characterizes the corresponding stochastic process. Roughly speaking the spectral density describes the "power" at a given spatial frequency $k$; for example, splines have $S(k) \propto k^{-\beta}$. The decay of $S(k)$ as $k$ increases is essential, as it provides a smoothing or damping out of high frequencies. Unfortunately non-stationary processes cannot be analyzed in exactly this fashion because the complex exponentials are not (in general) eigenfunctions of a non-stationary kernel. Instead, we must consider the eigenfunctions defined by $\int C(\boldsymbol{x}, \boldsymbol{x}')\phi(\boldsymbol{x}')d\boldsymbol{x}' = \lambda\phi(\boldsymbol{x})$. However, it may be possible to get some feel for the effect of a non-stationary covariance function by looking at the diagonal elements in its $2d$-dimensional Fourier transform, which correspond to the entries in power spectrum for stationary covariance functions.

### 3.4  Convergence of finite network priors to GPs

From general Central Limit Theorem results one would expect a rate of convergence of $H^{-1/2}$ towards a Gaussian process prior. How many units will be required in practice would seem to depend on the particular values of the weight-variance parameters. For example, for Gaussian transfer functions, $\sigma_m$ defines the radius over which we expect the process to be significantly different from zero. If this radius is increased (while keeping the variance of the basis functions $\sigma_g^2$ fixed) then naturally one would expect to need more hidden units in order to achieve the same level of approximation as before. Similar comments can be made for the sigmoidal case, depending on $(1 + 2\sigma_0^2)/2\sigma_I^2$.

I have conducted some experiments for the sigmoidal transfer function, comparing the predictive performance of a finite neural network with one input unit to the equivalent Gaussian process on data generated from the GP. The finite network simulations were carried out using a slightly modified version of Neal's MCMC Bayesian neural networks code (Neal, 1996) and the inputs were drawn from a

$N(0,1)$ distribution. The hyperparameter settings were $\sigma_I = 10.0$, $\sigma_0 = 2.0$, $\sigma_v = 1.189$ and $\sigma_b = 1.0$. Roughly speaking the results are that 100's of hidden units are required before similar performance is achieved by the two methods, although there is considerable variability depending on the particular sample drawn from the prior; sometimes 10 hidden units appears sufficient for good agreement.

## 4  Discussion

The work described above shows how to calculate the covariance function for sigmoidal and Gaussian basis functions networks. It is probable similar techniques will allow covariance functions to be derived analytically for networks with other kinds of basis functions as well; these may turn out to be similar in form to covariance functions already used in the Gaussian process literature.

In the derivations above the hyperparameters $\theta$ were fixed. However, in a real data analysis problem it would be unlikely that appropriate values of these parameters would be known. Given a prior distribution $P(\theta)$ predictions should be made by integrating over the posterior distribution $P(\theta|t) \propto P(\theta)P(t|\theta)$, where $P(t|\theta)$ is the likelihood of the training data $t$ under the model; $P(t|\theta)$ is easily computed for a Gaussian process. The prediction $\overline{y}(x)$ for test input $x$ is then given by

$$\overline{y}(x) = \int \hat{y}_\theta(x)P(\theta|D)d\theta \tag{14}$$

where $\hat{y}_\theta(x)$ is the predicted mean (as given by equation 4) for a particular value of $\theta$. This integration is not tractable analytically but Markov Chain Monte Carlo methods such as Hybrid Monte Carlo can be used to approximate it. This strategy was used in Williams and Rasmussen (1996), but for stationary covariance functions, not ones derived from Gaussian processes; it would be interesting to compare results.

### Acknowledgements

I thank David Saad and David Barber for help in obtaining the result in equation 11, and Chris Bishop, Peter Dayan, Ian Nabney, Radford Neal, David Saad and Huaiyu Zhu for comments on an earlier draft of the paper. This work was partially supported by EPSRC grant GR/J75425, "Novel Developments in Learning Theory for Neural Networks".

## Footnotes

[1]For large $n$, various approximations to the exact solution which avoid the inversion of an $n \times n$ matrix are available.

[2]For notational convenience we suppress the $x$-dependence of the posterior.

[3]Introduce a dummy parameter $\lambda$ to make the first term in the integrand $\Phi(\lambda u^T \tilde{x})$. Differentiate the integral with respect to $\lambda$ and then use integration by parts. Finally recognize that $dV_{\text{erf}}/d\lambda$ is of the form $(1 - \theta^2)^{-1/2} d\theta/d\lambda$ and hence obtain the $\sin^{-1}$ form of the result, and evaluate it at $\lambda = 1$.

[4]Note that this would require $\omega^2 \to \infty$ and hence the Central Limit Theorem would no longer hold, i.e. the process would be non-Gaussian.

## References

Cressie, N. A. C. (1993). *Statistics for Spatial Data*. Wiley.

Hornik, K. (1993). Some new results on neural network approximation. *Neural Networks* **6** (8), 1069–1072.

Journel, A. G. and C. J. Huijbregts (1978). *Mining Geostatistics*. Academic Press.

MacKay, D. J. C. (1992). A Practical Bayesian Framework for Backpropagation Networks. *Neural Computation* **4(3)**, 448–472.

Neal, R. M. (1996). *Bayesian Learning for Neural Networks*. Springer. Lecture Notes in Statistics 118.

Parzen, E. (1962). *Stochastic Processes*. Holden-Day.

Poggio, T. and F. Girosi (1990). Networks for approximation and learning. *Proceedings of IEEE* **78**, 1481–1497.

Wahba, G. (1990). *Spline Models for Observational Data*. Society for Industrial and Applied Mathematics. CBMS-NSF Regional Conference series in applied mathematics.

Whittle, P. (1963). *Prediction and regulation by linear least-square methods*. English Universities Press.

Williams, C. K. I. and C. E. Rasmussen (1996). Gaussian processes for regression. In D. S. Touretzky, M. C. Mozer, and M. E. Hasselmo (Eds.), *Advances in Neural Information Processing Systems 8*, pp. 514–520. MIT Press.
